# Measure Based Regularization

**Olivier Bousquet, Olivier Chapelle, Matthias Hein**
Max Planck Institute for Biological Cybernetics, 72076 Tübingen, Germany
{*first.last*}*@tuebingen.mpg.de*

## Abstract

We address in this paper the question of how the knowledge of
the marginal distribution $P(x)$ can be incorporated in a learning
algorithm. We suggest three theoretical methods for taking into
account this distribution for regularization and provide links to
existing graph-based semi-supervised learning algorithms. We also
propose practical implementations.

## 1 Introduction

Most existing learning algorithms perform a trade-off between fit of the data and
'complexity' of the solution. The way this complexity is defined varies from one
algorithm to the other and is usually referred to as a prior probability or a regular-
izer. The choice of this term amounts to having a preference for certain solutions
and there is no a priori best such choice since it depends on the learning problem
to be addressed. This means that the right choice should be dictated by prior
knowledge or assumptions about the problem or the class of problems to which
the algorithm is to be applied. Let us consider the binary classification setting. A
typical assumption that is (at least implicitly) used in many learning algorithms is
the following

> *Two points that are close in input space should have the same label.*

One possible way to enforce this assumption is to look for a decision function which
is consistent with the training data and which does not change too much between
neighboring points. This can be done in a regularization setting, using the Lips-
chitz norm as a regularizer. For differentiable functions, the Lipschitz norm of a
function is the supremum of the norm of the gradient. It is thus natural to consider
algorithms of the form

$$\min_f \ \sup_{\mathbf{x}} \|\nabla f(\mathbf{x})\| \quad \text{under constraints} \quad y_i f(\mathbf{x}_i) \geq 1. \tag{1}$$

Performing such a minimization on the set of linear functions leads to the maximum
margin solution (since the gradient $\mathbf{x} \mapsto \langle \mathbf{w}, \mathbf{x} \rangle$ is $\mathbf{w}$), whereas the 1-nearest neighbor
decision function is one of the solutions of the above optimization problem when
the set of functions is unconstrained [13].
Although very useful because widely applicable, the above assumption is sometimes
too weak. Indeed, most 'real-world' learning problems have more structure than
what this assumption captures. For example, most data is located in regions where

the label is constant (clusters) and regions where the label is not well-defined are typically of low density. This can be formulated via the so-called *cluster assumption*:

> *Two points that are connected by a line that goes through high density regions should have the same label*

Another related way of stating this assumption is to say that the decision boundary should lie in regions of *low density*.

Our goal is to propose possible implementations of this assumption. It is important to notice that in the context of supervised learning, the knowledge of the joint probability $P(\mathbf{x}, y)$ is enough to achieve perfect classification (taking $\arg\max_y P(\mathbf{x}, y)$ as decision function), while in semi-supervised learning, even if one knows the distribution $P(\mathbf{x})$ of the instances, there is no unique or optimal way of using it. We will thus try to propose a principled approach to this problem. A similar attempt was made in [10] but in a probabilistic context, where the decision function was modeled by a conditional probability distribution, while here we consider arbitrary real-valued functions and use the standard regularization approach.

We will use three methods for obtaining regularizers that depend on the distribution $P(\mathbf{x})$ of the data. In section 2 we suggest to modify the regularizer in a general way by weighting it with the data density. Then in section 3 we adopt a geometric approach where we suggest to modify the distances in input space (in a local manner) to take into account the density (i.e. we stretch or blow up the space depending on the density). The third approach presented in section 4 builds on spectral methods. The idea is to look for the analogue of graph-based spectral methods when the amount of available data is infinite. We show that these three approaches are related in various ways and in particular we clarify the asymptotic behavior of graph-based regularization. Finally, in section 5 we give a practical method for implementing one of the proposed regularizers and show its application on a toy problem.

## 2   Density based regularization

The first approach we propose is to start with a gradient-based regularizer like $\|\nabla f\|$ which penalizes large variations of the function. Now, to implement the cluster assumption one has to penalize more the variations of the function in high density regions and less in low density regions. A natural way of doing this is to replace $\|\nabla f\|$ by $\|p\nabla f\|$ where $p$ is the density of the marginal distribution $P$. More generally, instead of the gradient, one can can consider a regularization map $L : \mathbb{R}^{\mathcal{X}} \mapsto (\mathbb{R}^+)^{\mathcal{X}}$, where $L(f)(\mathbf{x})$ is a measure of the smoothness of the function $f$ at the point $\mathbf{x}$, and then consider the following regularization term

$$\Omega(f) = \| \, L(f)\chi(p) \, \|, \tag{2}$$

where $\chi$ is a strictly increasing function.

An interesting case is when the norm in (2) is chosen as the $L_2$ norm. Then, $\Omega(f)$ can be the norm of a Reproducing Kernel Hilbert Space (RKHS), which means that there exist an Hilbert space $\mathcal{H}$ and a kernel function $k : \mathcal{X}^2 \mapsto \mathbb{R}$ such that

$$\sqrt{\langle f, f \rangle_{\mathcal{H}}} = \Omega(f) \;\; \text{and} \;\; \langle f, k(\mathbf{x}, \cdot) \rangle_{\mathcal{H}} = f(\mathbf{x}). \tag{3}$$

The reason for using an RKHS norm is the so-called representer theorem [5]: the function minimizing the corresponding regularized loss can be expressed as a linear combination of the kernel function evaluated at the labeled points.

However, it is not straightforward to find the kernel associated with an RKHS norm. In general, one has to solve equation (3). For instance, in the case $L(f) = (f^2 + \|\nabla f\|^2)^{1/2}$ and without taking the density into account ($\chi = 1$), it has been shown in [3] that the corresponding kernel is the Laplacian one, $k(\mathbf{x}, \mathbf{y}) = \exp(-\|\mathbf{x} - \mathbf{y}\|_{L_1})$ with associated inner product $\langle f, g \rangle_{\mathcal{H}} = \langle f, g \rangle_{L_2} + \langle \nabla f, \nabla g \rangle_{L_2}$. Taking the density into account, this inner product becomes

$$\langle f, g \rangle_{\mathcal{H}} = \langle f, \chi^2(p)g \rangle_{L_2} + \langle \nabla f, \chi^2(p)\nabla g \rangle_{L_2}.$$

Plugging $g = k(\mathbf{x}, .)$ in above and expressing that (3) should be valid for all $f \in \mathcal{H}$, we find that $k$ must satisfy

$$\chi^2(p)k(\mathbf{x}, .) - \nabla(\chi^2(p)\nabla k(\mathbf{x}, .)) = \delta(\mathbf{x} - .),$$

where $\delta$ is the Dirac delta function. However, solving this differential equation is not an easy task for arbitrary $p$.

Since finding the kernel function associated to a regularizer is, in general, a difficult problem, we propose to perform the minimization of the regularized loss on a fixed set of basis functions, i.e. $f$ is expressed as a linear combination of functions $\varphi_i$.

$$f(\mathbf{x}) = \sum_{i=1}^{l} \alpha_i \varphi_i(\mathbf{x}) + b. \tag{4}$$

We will present in section 5 a practical implementation of this approach.

## 3 Density based change of geometry

We now try to adopt a geometric point of view. First we translate the cluster assumption into a geometric statement, then we explore how to enforce it by changing the geometry of our underlying space. A similar approach was recently proposed by Vincent and Bengio [12]. We will see that there exists such a change of geometry which leads to the same type of regularizer that was proposed in section 2.

Recall that the cluster assumption states that points are likely to be in the same class if they can be connected by a path through high density regions. Naturally this means that we have to weight paths according to the density they are going through. This leads to introducing a new distance measure on the input space (typically $\mathbb{R}^d$) defined as the length of the shortest weighted path connecting two points. With this new distance, we simply have to enforce that close points have the same label (we thus recover the standard assumption).

Let us make this more precise. We consider the euclidean space $\mathbb{R}^d$ as a flat Riemannian manifold with metric tensor $\delta$, denoted by $(\mathbb{R}^n, \delta)$. A Riemannian manifold $(\mathcal{M}, g)$ is also a metric space with the following path (or geodesic) distance:

$$d(x, y) = \inf_{\gamma} \{L(\gamma) | \gamma : [a, b] \to \mathcal{M}, \gamma(a) = x, \gamma(b) = y\}$$

where $\gamma$ is a piecewise smooth curve and $L(\gamma)$ is the length of the curve given by

$$L(\gamma) = \int_a^b \sqrt{g_{ij}(\gamma(t))\dot{\gamma}^i\dot{\gamma}^j}\,dt \tag{5}$$

We now want to change the metric $\delta$ of $\mathbb{R}^d$ such that the new distance is the weighted path distance corresponding to the cluster assumption. The only information we have is the local density $p(x)$, which is a scalar at every point and as such can only lead to an isotropic transformation in the tangent space $T_x\mathcal{M}$. Therefore we consider the following conformal transformation of the metric $\delta$

$$\delta_{ij} \to g_{ij} = \frac{1}{\chi(p(x))}\delta_{ij} \tag{6}$$

where $\chi$ is a strictly increasing function. We denote by $(\mathbb{R}^d, g)$ the distorted euclidean space. Note that this kind of transformation also changes the volume element $\sqrt{g}dx^1 \ldots dx^d$, where $g$ is the determinant of $g_{ij}$.

$$dx^1 \ldots dx^d \rightarrow \sqrt{g}dx^1 \ldots dx^d = \frac{1}{\chi(p)^{d/2}}dx^1 \ldots dx^d \qquad (7)$$

In the following we will choose $\chi(x) = x$, which is the simplest choice which gives the desired properties.

The distance structure of the transformed space implements now the cluster assumption, since we see from (5) that all paths get weighted by the inverse density. Therefore we can use any metric based classification method and it will automatically take into account the density of the data. For example the nearest neighbor classifier in the new distance is equivalent to the Lipschitz regularization (1) weighted with the density proposed in the last section.

However, implementing such a method requires to compute the geodesic distance in $(\mathbb{R}^d, g)$, which is non trivial for arbitrary densities $p$. We suggest the following approximation which is similar in spirit to the approach in [11].

Since we have a global chart of $\mathbb{R}^d$ we can give for each neighborhood $B_\epsilon(x)$ in the euclidean space the following upper and lower bounds for the geodesic distance:

$$\inf_{z \in B_\epsilon(x)} \sqrt{\frac{1}{p(z)}} \|x - y\| \leq d(x, y) \leq \sup_{z \in B_\epsilon(x)} \sqrt{\frac{1}{p(z)}} \|x - y\|, \quad \forall y \in B_\epsilon(x) \qquad (8)$$

Then we choose a real $\epsilon$ and set for each $x$ the distance to all points in a $p(x)^{-1/2}\epsilon$-neighborhood of $x$ as $d(x, y) = p(\frac{x+y}{2})^{-1/2}\|x - y\|$. The geodesic distance can then be approximated by the shortest path along the obtained graph.

We now show the relationship to the the regularization based approach of the previous section. We denote by $\|\cdot\|_{L_2(\mathbb{R}^d, g, \Sigma)}$ the $L_2$ norm in $(\mathbb{R}^d, g)$ with respect to the measure $\Sigma$ and by $\mu$ the standard Lebesgue measure on $\mathbb{R}^d$. Let us consider the regularizer $\|\nabla f\|^2_{L_2(\mathbb{R}^d, \delta, \mu)}$ which is the standard $L_2$ norm of the gradient. Now modifying this regularizer according to section 2 (by changing the underlying measure) gives $S(f) = \|\nabla f\|^2_{L_2(\mathbb{R}^d, \delta, P)}$. On the distorted space $(\mathbb{R}^d, g)$ we keep the Lebesgue measure $\mu$ which can be done by integrating on the manifold with respect to the density $\sigma = \frac{1}{\sqrt{g}} = p^{d/2}$, which cancels then with the volume element $\sigma\sqrt{g}dx^1 \ldots dx^d = dx^1 \ldots dx^d$. Since we have on $(\mathbb{R}^d, g)$, $\|\nabla f\|^2 = p(x)\delta^{ij}\frac{\partial f}{\partial x^i}\frac{\partial f}{\partial x^j}$ we get equivalence of $S(f)$.

$$S(f) = \|\nabla f\|^2_{L_2(\mathbb{R}^d, \delta, P)} = \int_{\mathbb{R}^d} p(x)\delta^{ij}\frac{\partial f}{\partial x^i}\frac{\partial f}{\partial x^j}dx^1 \ldots dx^d = \|\nabla f\|^2_{L_2(\mathbb{R}^d, g, \mu)} \qquad (9)$$

*This shows that modifying the measure and keeping the geometry, or modifying the geometry and keeping the Lebesgue measure leads to the same regularizer $S(f)$.* However, there is a structural difference between the spaces $(\mathbb{R}^d, \delta, P)$ and $(\mathbb{R}^d, g, \mu)$ even if $S(f)$ is the same. Indeed, for regularization operators corresponding to higher order derivatives the above correspondence is not valid any more.

## 4   Link with Spectral Techniques

Recently, there has been a lot of interest in spectral techniques for non linear dimension reduction, clustering or semi-supervised learning. The general idea of these approaches is to construct an adjacency graph on the (unlabeled) points whose weights are given by a matrix $W$. Then the first eigenvectors of a modified version

of $W$ give a more suitable representation of the points (taking into account their manifold and/or cluster structure). An instance of such an approach and related references are given in [1] where the authors propose to use the following regularizer

$$\frac{1}{2} \sum_{i,j=1}^{m} (f_i - f_j)^2 W_{ij} = \mathbf{f}^\top (D - W)\mathbf{f}, \tag{10}$$

where $f_i$ is the value of the function at point $\mathbf{x}_i$ (the index ranges over labeled and unlabeled points), $D$ is a diagonal matrix with $D_{ii} = \sum_j W_{ij}$ and $W_{ij}$ is chosen as a function of the distance between $\mathbf{x}_i$ and $\mathbf{x}_j$, for example $W_{ij} = K(\|\mathbf{x}_i - \mathbf{x}_j\|/t)$. Given a sample $\mathbf{x}_1, \ldots, \mathbf{x}_m$ of $m$ i.i.d. instances sampled according to $P(\mathbf{x})$, it is possible to rewrite (10) after normalization as the following random variable

$$U_f = \frac{1}{2m(m-1)} \sum_{i,j} (f(\mathbf{x}_i) - f(\mathbf{x}_j))^2 K(\|\mathbf{x}_i - \mathbf{x}_j\|/t).$$

Under the assumption that $f$ and $K$ are bounded, the result of [4] (see Inequality (5.7) in this paper, which applies to U-statistics) gives

$$\mathbb{P}\left[U_f \geq \mathbb{E}\left[U_f\right] + t\right] \leq e^{-mt^2/C^2},$$

where $C$ is a constant which does not depend on $n$ and $t$. This shows that for each fixed function, the normalized regularizer $U_f$ converges towards its expectation when the sample size increases. Moreover, one can check that

$$\mathbb{E}[U_f] = \frac{1}{2} \int \int (f(\mathbf{x}) - f(\mathbf{y}))^2 K(\|\mathbf{x} - \mathbf{y}\|/t) dP(\mathbf{x}) dP(\mathbf{y}). \tag{11}$$

This is the term that should be used as a regularizer if one knows the whole distribution since it is the limit of $(10)^1$.

The following proposition relates the regularizer (11) to the one defined in (2).

**Proposition 4.1** *If $p$ is a density which is Lipschitz continuous and $K$ is a continuous function on $\mathbb{R}^+$ such that $x^{2+d}K(x) \in L_2$, then for any function $f \in C^2(\mathbb{R}^d)$ with bounded hessian*

$$\lim_{t \to 0} \frac{d}{C\, t^{2+d}} \int \int (f(\mathbf{x}) - f(\mathbf{y}))^2 K(\|\mathbf{x} - \mathbf{y}\|/t) p(\mathbf{x}) p(\mathbf{y}) d\mathbf{x} d\mathbf{y} \tag{12}$$

$$= \int \|\nabla f(\mathbf{x})\|^2 p^2(\mathbf{x}) d\mathbf{x}, \tag{13}$$

*where $C = \int_{\mathbb{R}^d} \|\mathbf{x}\|^2 K(\|\mathbf{x}\|) d\mathbf{x}$.*

**Proof:** Let's fix $\mathbf{x}$. Writing a Taylor-Lagrange expansion of $f$ and $p$ around $\mathbf{x}$ in terms of $\mathbf{h} = (\mathbf{y} - \mathbf{x})/t$ gives

$$\int (f(\mathbf{x}) - f(\mathbf{y}))^2 K\left(\frac{\|\mathbf{x} - \mathbf{y}\|}{t}\right) p(\mathbf{y}) d\mathbf{y}$$

$$= \int (t \langle \nabla f(\mathbf{x}), \mathbf{h}\rangle + O(t^2 \|\mathbf{h}\|^2))^2 K(\|\mathbf{h}\|)(p(\mathbf{x}) + O(t \|\mathbf{h}\|)) t^d d\mathbf{h}$$

$$= t^{d+2} p(\mathbf{x}) \int \langle \nabla f(\mathbf{x}), \mathbf{h}\rangle^2 K(\|\mathbf{h}\|) d\mathbf{h} + O(t^{d+3}), \tag{14}$$

To conclude the proof, we rewrite this last integral as $\nabla f(\mathbf{x})^\top \left( \int \mathbf{h}\mathbf{h}^\top K(\|\mathbf{h}\|)d\mathbf{h} \right) \nabla f(\mathbf{x}) = \|\nabla f(\mathbf{x})\|^2 \frac{C}{d}$. The last equality comes from the fact that, by symmetry considerations, $\int \mathbf{h}\mathbf{h}^\top K(\|\mathbf{h}\|)d\mathbf{h}$ is equal to a constant (let's call it $C_2$) times the identity matrix and this constant can be computed by $C_2 d = \text{trace}\left( \int \mathbf{h}\mathbf{h}^\top K(\|\mathbf{h}\|)d\mathbf{h} \right) = \text{trace}\left( \int \mathbf{h}^\top \mathbf{h} K(\|\mathbf{h}\|)d\mathbf{h} \right) = C.$ □

Note that different $K$ lead to different affinity matrices: if we choose $K(x) = \exp(-x^2/2)$, we get a gaussian RBF affinity matrix as used in [7], whereas $K(x) = 1_{x \leq 1}$ leads to an unweighted neighboring graph (at size $t$) [1].
So we have proved that if one takes the limit of the regularizer (10) when the sample size goes to infinity and the scale parameter $t$ goes to 0 (with appropriate scaling), one obtains the regularizer

$$\int \|\nabla f(\mathbf{x})\|^2 \, p^2(\mathbf{x})d\mathbf{x} = \left\langle f, \nabla^* D_p^2 \nabla f \right\rangle,$$

where $\nabla^*$ is the adjoint of $\nabla$, $D_p$ is the diagonal operator that maps $f$ to $pf$ and $\langle .,. \rangle$ is the inner product in $L_2$.
In [2], the authors investigated the limiting behavior of the regularizer $D - W$ obtained from the graph and claimed that this is the empirical counterpart of the Laplace operator defined on the manifold. However, this is true only if the distribution is uniform on the manifold. We have shown that, in the general case, the continuous equivalent of the graph Laplacian is $\nabla^* D_p^2 \nabla$.

## 5 Practical Implementation and Experiments

As mentioned in section 2, it is difficult in general to find the kernel associated with a given regularizer and instead, we decided to minimize the regularized loss on a fixed basis of functions $(\varphi_i)_{1 \leq i \leq l}$, as expressed by equation (4).

The regularizer we considered is of the form (2) and is,

$$\Omega(f) = \| \, \|\nabla f\| \, \sqrt{p} \, \|_{L_2}^2 = \int \nabla f(\mathbf{x}) \cdot \nabla f(\mathbf{x}) p(\mathbf{x})d\mathbf{x}.$$

Thus, the coefficients $\boldsymbol{\alpha}$ and $b$ in expansion (4) are found by minimizing the following convex regularized functional

$$\underbrace{\frac{1}{n}\sum_{i=1}^n \ell(f(\mathbf{x}_i), y_i)}_{R_{emp}(f)} + \lambda \underbrace{\sum_{i,j=1}^l \alpha_i \alpha_j \int \nabla\varphi_i(\mathbf{x}) \cdot \nabla\varphi_j(\mathbf{x}) p(\mathbf{x})d\mathbf{x}}_{\|L(f)\sqrt{p}\|_{L_2}^2}. \tag{15}$$

Introducing the $l \times l$ matrix $H_{ij} = \int \nabla\varphi_i(\mathbf{x}) \cdot \nabla\varphi_j(\mathbf{x})p(\mathbf{x})d\mathbf{x}$ and the $n \times l$ matrix $K$ with $K_{ij} = \varphi_j(\mathbf{x}_i)$, the minimization of the functional (15) is equivalent to the following one for the standard $L_1$-SVM loss:

$$\min_{\boldsymbol{\alpha},b} \quad \boldsymbol{\alpha}^\top H \boldsymbol{\alpha} + C \sum_{i=1}^n \xi_i$$

under constraints $\forall i, \quad y_i(\sum_{j=1}^l K_{ij}\alpha_j + b) \geq 1 - \xi_i$. The dual formulation of this optimization problem turns out to be the standard SVM one with a modified kernel function (see also [9]):

$$\max_{\beta} \sum_{i=1}^n \beta_i - \frac{1}{2}\sum_{i,j=1}^n \beta_i \beta_j y_i y_j L_{ij},$$

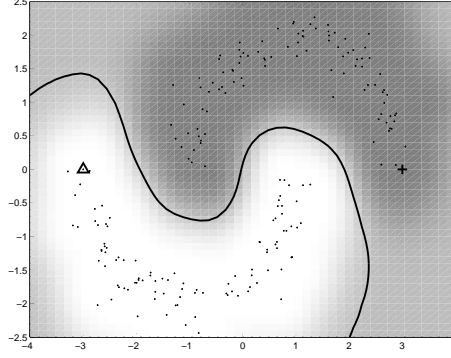

Figure 1: Two moons toy problem: there are 2 labeled points (the cross and the triangle) and 200 unlabeled points. The gray level corresponds to the output of the function. The function was expanded on all unlabeled points (m=200 in (4)) and the widths of the gaussians have been chosen as $\sigma = 0.5$ and $\sigma_p = 0.05$.

under constraints $0 \leq \beta_i \leq C$ and $\sum \beta_i y_i = 0$, with $L = KH^{-1}K^{\top}$.

Once the vector $\boldsymbol{\beta}$ has been found, the coefficients $\boldsymbol{\alpha}$ of the expansion are given by

$$\boldsymbol{\alpha} = H^{-1}K^{\top}\operatorname{diag}(Y)\boldsymbol{\beta}.$$

In order to calculate the $H_{ij}$, one has to compute an integral. From now on, we consider a special case where this integral can be computed analytically:

- The basis functions are gaussian RBF, $\varphi_i(\mathbf{x}) = \exp\left(-\frac{\|\mathbf{x}-\mathbf{x}_i\|^2}{2\sigma^2}\right)$, where the points $\mathbf{x}_1, \ldots, \mathbf{x}_l$ can be chosen arbitrarily. We decided to take the unlabeled points (or a subset of them) for this expansion.

- The marginal density $p$ is estimated using a Parzen window with a Gaussian kernel, $p(\mathbf{x}) = \frac{1}{m}\sum_{i=1}^{m}\exp\left(-\frac{\|\mathbf{x}-\mathbf{x}_i\|^2}{2\sigma_p^2}\right)$.

Defining $h = 1/\sigma^2$ and $h_p = 1/\sigma_p^2$, this integral turns out to be, up to an irrelevant constant factor,

$$H_{ij} = \sum_{k=1}^{m}\exp\left(-\frac{h^2}{2h+h_p}\frac{\|\mathbf{x}_i-\mathbf{x}_j\|^2}{2} - \frac{hh_p}{2h+h_p}\frac{\|\mathbf{x}_i-\mathbf{x}_k\|^2 + \|\mathbf{x}_j-\mathbf{x}_k\|^2}{2}\right)$$
$$\left(h_p^2(\mathbf{x}_k-\mathbf{x}_i)\cdot(\mathbf{x}_k-\mathbf{x}_j) - h(h+h_p)(\mathbf{x}_i-\mathbf{x}_j)^2 + d(2h+h_p)\right),$$

where $d$ is the dimension of the input space.

After careful dataset selection [6], we considered the two moons toy problem (see figure 1). On this 2D example, the regularizer we suggested implements perfectly the cluster assumption: the function is smooth on high density regions and the decision boundary lies in a low density region.

We also tried some real world experiments but were not successful. The reason might be that in dimension more than 2, the gradient does not yield a suitable regularizer: there exists non continuous functions whose regularizer is 0. To avoid this, from the Sobolev embedding lemma, we consider derivatives of order at least $d/2$. More specifically, we are currently investigating the regularizer associated with

a Gaussian kernel of width $\sigma_r$ [8, page 100],

$$\sum_{p=1}^{\infty} \frac{\sigma_r^{2p}}{p!2^p} \int \left\| \nabla^p f(\mathbf{x}) \right\|^2 p(\mathbf{x}) d\mathbf{x}, \quad \text{with } \nabla^{2p} \equiv \Delta^p.$$

## 6   Conclusion

We have tried to make a first step towards a theoretical framework for semi-supervised learning. Ideally, this framework should be based on general principles which can then be used to derive new heuristics or justify existing ones.

One such general principle is the cluster assumption. Starting from the assumption that the distribution $P(\mathbf{x})$ of the data is known, we have proposed several ideas to implement this principle and shown their relationships. In addition, we have shown the relationship to the limiting behavior of an algorithm based on the graph Laplacian.

We believe that this topic deserves further investigation. From a theoretical point of view, other types of regularizers, involving, for example, higher order derivatives should be studied. Also from a practical point of view, we should derive efficient algorithms from the proposed ideas, especially by obtaining finite sample approximations of the limit case where $P(\mathbf{x})$ is known.

## Footnotes

[1] We have shown that the convergence of $U_f$ towards $\mathbb{E}[U_f]$ happens for each fixed $f$ but this convergence can be uniform over a set of functions, provided this set is small enough.

## References

[1] M. Belkin and P. Niyogi. Laplacian eigenmaps for dimensionality reduction and data representation. *Neural Computation*, 15(6):1373–1396, 2003.

[2] M. Belkin and P. Niyogi. Semi-supervised learning on manifolds. *Machine Learning journal*, 2003. to appear.

[3] F. Girosi, M. Jones, and T. Poggio. Priors, stabilizers and basis functions: From regularization to radial, tensor and additive splines. Technical Report Artificial Intelligence Memo 1430, Massachusetts Institute of Technology, 1993.

[4] W. Hoeffding. Probability inequalities for sums of bounded random variables. *Journal of the American Statistical Association*, 58:13–30, 1963.

[5] G. Kimeldorf and G. Wahba. Some results on tchebychean spline functions. *Journal of Mathematics Analysis and Applications*, 33:82–95, 1971.

[6] Doudou LaLoudouana and Mambobo Bonouliqui Tarare. Data set selection. In *Advances in Neural Information Processing Systems*, volume 15, 2002.

[7] A. Y. Ng, M. I. Jordan, and Y. Weiss. On spectral clustering: Analysis and an algorithm. In *Advances in Neural Information Processing Systems*, volume 14, 2001.

[8] B. Schölkopf and A. Smola. *Learning with kernels*. MIT Press, Cambridge, MA, 2002.

[9] A. Smola and B. Scholkopf. On a kernel-based method for pattern recognition, regression, approximation and operator inversion. *Algorithmica*, 22:211–231, 1998.

[10] M. Szummer and T. Jaakkola. Information regularization with partially labeled data. In *Advances in Neural Information Processing Systems*, volume 15. MIT Press, 2002.

[11] J. B. Tenenbaum, V. de Silva, and J. C. Langford. A global geometric framework for nonlinear dimensionality reduction. *Science*, 290(5500):2319–2323, 2000.

[12] P. Vincent and Y. Bengio. Density-sensitive metrics and kernels. Presented at the Snowbird Learning Workshop, 2003.

[13] U. von Luxburg and O. Bousquet. Distance-based classification with lipschitz functions. In *Proceedings of the 16th Annual Conference on Computational Learning Theory*, 2003.
